# Blind source separation for over-determined delayed mixtures

**Lars Omlor, Martin Giese**[*]
Laboratory for Action Representation and Learning
Department of Cognitive Neurology,
Hertie Institute for Clinical Brain Research
University of Tübingen, Germany

## Abstract

Blind source separation, i.e. the extraction of unknown sources from a set of given signals, is relevant for many applications. A special case of this problem is *dimension reduction*, where the goal is to approximate a given set of signals by superpositions of a minimal number of sources. Since in this case the signals outnumber the sources the problem is over-determined. Most popular approaches for addressing this problem are based on purely linear mixing models. However, many applications like the modeling of acoustic signals, EMG signals, or movement trajectories, require temporal shift-invariance of the extracted components. This case has only rarely been treated in the computational literature, and specifically for the case of dimension reduction almost no algorithms have been proposed. We present a new algorithm for the solution of this problem, which is based on a time-frequency transformation (Wigner-Ville distribution) of the generative model. We show that this algorithm outperforms classical source separation algorithms for linear mixtures, and also a related method for mixtures with delays. In addition, applying the new algorithm to trajectories of human gaits, we demonstrate that it is suitable for the extraction of spatio-temporal components that are easier to interpret than components extracted with other classical algorithms.

## 1   Introduction

Blind source separation techniques, such as Independent Components Analysis (ICA), have received great interest in many domains including neuroscience [3; 19; 2], machine learning [12; 11], and speech and signal processing [25].

A variety of algorithms have been proposed for different types of mixing models. Many studies have focused on *instantaneous mixing*, where target signals are modeled by the linear superposition of a number of source signals separately for each point in time. Another set of studies has treated *convolutive mixing*, where signals result from the superposition of filtered source signals (see [9] and [6] for review). Much less explored are algorithms for *anechoic mixing*. In this case, signals are approximated by linear combinations of source signals with time delays. Classical cases of anechoic mixing arise in electrical engineering, when signals from multiple antennas are received asynchronously, or in acoustics when sound signals are recorded with multiple microphones resulting in different running times. A few algorithms have been proposed for the solution of *under-determined* anechoic mixing problems, where the number of sources exceeds the number of signals [8; 4; 25; 22]. A method that treats the case of equal numbers of signals and sources, which is based on joint diagonalization of spectral matrices, has been proposed by Yeredor [24]. Almost no work exists on *over-determined* anechoic mixing problems, where the number of source signals is smaller than the number of original signals – the case that is most important for dimension reduction problems. Most

---

[*]WWW home page: http://www.uni-tuebingen.de/uni/knv/arl/index.html

existing methods for the solution of under-determined problems cannot be transferred to the over-determined case, because they involve additional assumptions about the data (e.g. specific spatial structure [20]) or the solution (e.g. sparseness [4]). One approach employed for under-determined anechoic mixtures is based on the assumption of small delays and a linearization of the mixture model [5]. While this original method cannot be transferred to our problem, since it requires additional assumptions about the spatial structure of the data, preliminary work in [1] applies the same basic approximation for the over-determined case.

In this paper we present a new algorithm for the solution of the over-determined anechoic mixing problem, which makes no further assumptions about the size of the delays. The proposed method is derived by applying methods from stochastic time-frequency analysis. We tested the novel algorithm with two different test data sets, human movement trajectories and synthetic mixtures of acoustic signals. We demonstrate that the method results in more accurate solutions with fewer sources than classical methods (like PCA and normal ICA) for instantaneous mixing. Also, we demonstrate that our algorithm outperforms the SOBIDS algorithm in [1] for anechoic mixtures. In addition, we demonstrate that the method seems suitable for the extraction of biologically meaningful components from human movement data.

## 2 Source separation for over-determined delayed mixtures

### 2.1 Delayed mixture problem

In the following we assume that $m$ signals $x_i(t)$, $1 \leq i \leq m$ have been observed. These signals are approximated by a linear combination of $n$ source signals $s_j(t)$ with $1 \leq j \leq n$, with temporal delays $\tau_{ij}$. In the case of anechoic mixing signals and sources obey the relationship:

$$x_i(t) = \sum_{j=1}^{n} \alpha_{ij} \cdot s_j(t - \tau_{ij}) \quad i = 1, \cdots, m \tag{1}$$

In the over-determined case the signals outnumber the sources, i.e. $m \geq n$.
Equation (1) is a special case of a convolutive mixture problem, where the filter kernels are given by delta functions. However, the treatment as general deconvolution problem would neglect the special structure of the convolutive kernel that is given by a weighted sum of delta pulses:

$$x_i = \sum_{j=1}^{n} (\alpha_{ij}\delta(t - \tau_{ij})) * s_j \quad i = 1, \cdots, m \tag{2}$$

Nevertheless, this formulation suggests a treatment of this problem exploiting the framework of harmonic analysis. Since normal Fourier transformation of equation (1) results in frequency-dependent mixtures of complex phase terms, time-frequency analysis turns out to be a more appropriate framework for the separation of sources in the above mixture models.

### 2.2 Wigner Ville Spectrum

In signal processing and acoustics a variety of time-frequency representations have been proposed, ranging from linear and multilinear to nonlinear transformations. Due to their close connections to energy and correlation measures, specifically bilinear or quadratic distributions seem very appealing. A very popular quadratic representation is the Wigner distribution, and its modifications that are included in Cohen's class [7].
While the Wigner Ville Spectrum is usually defined as a deterministic integral transform, it can also be extended for the analysis of (nonstationary) random processes resulting in the following definition for the Wigner Ville spectrum (WVS): Assuming that $x(t)$ is a random process and $x^*(t) = \bar{x}(-t)$ the reversed conjugated process, the WVS can be defined as [16]:

$$W_x(t,\omega) = \int_{\tau} E\left\{ x\left(t + \frac{\tau}{2}\right) x^*\left(t - \frac{\tau}{2}\right) \right\} e^{-2\pi \mathrm{i}\omega\tau} d\tau \tag{3}$$

The WVS is basically a 2-D function defined over the time-frequency plain, which can loosely be interpreted as a time-frequency distribution of the mean energy of $x(t)$. The definition (3) implies many useful properties [16], three of which are particularly useful for the following derivation. If $\mathcal{F}$ denotes the Fourier Transform and $T_\tau$, $M_f$ the time and frequency shift operators, e.g.

$(T_\tau M_f x)(t) := e^{-2\pi \mathrm{i} f(t-\tau)} x(t-\tau)$, then the WVS has the following properties:

1. Time-frequency shift covariance:
$$W_{(T_\tau M_f x)}(t,\omega) = W_x(t-\tau, \omega - f) \tag{4}$$

2. Marginal properties:
$$\int W_x(t,\omega)dt = E\{|\mathcal{F}x|^2\} \tag{5}$$

$$\int W_x(t,\omega)df = E\{|x|^2\} \tag{6}$$

3. Mean group delay:
$$t_x(\omega) := \frac{\int t W_x(t,\omega)dt}{\int W_x(t,\omega)dt} \tag{7}$$

Since the group delay (7) is not uniquely defined in the stochastic case [15], the last property gives a natural definition. Consistent with the standard definition for deterministic signals, the group delay for the deterministic case can be rewritten:

$$\frac{-1}{2\pi} \frac{\partial}{\partial \omega} \arg((\mathcal{F}x)(\omega)) = t_x(\omega) = \frac{\int t W_x(t,\omega)dt}{\int W_x(t,\omega)dt}$$

.

## 2.3 Application to the delayed mixture model

The original mixture model can be rewritten in the time-frequency domain by computing the WVS of both sides of (1):

$$W_{x_i}(t,\omega) = \int \mathrm{E}\left\{ \sum_{j,k=1}^n \alpha_{ij}\overline{\alpha_{ik}} s_j(t + \frac{\tau}{2} - \tau_{ij}) s_k^*(t - \frac{\tau}{2} - \tau_{ik}) \right\} e^{-2\pi \mathrm{i}\omega\tau} d\tau$$

$$= \sum_{j,k=1}^n \alpha_{ij}\overline{\alpha_{ik}} \int \mathrm{E}\left\{ s_j(t + \frac{\tau}{2} - \tau_{ij}) s_k^*(t - \frac{\tau}{2} - \tau_{ik}) \right\} e^{-2\pi \mathrm{i}\omega\tau} d\tau \tag{8}$$

Assuming that the source signals $s_j$ are statistically independent equation (8) can be further simplified, since all cross terms of the form $E\{s_i s_j\}$ with $i \neq j$ will vanish (assuming additionally $E\{s_i\} = 0$). This together with the shift covariance of the distribution leads to the central equation:

$$W_{x_i}(t,\omega) = \sum_j^n |\alpha|_{ij}^2 W_{s_j}(t - \tau_{ij}, \omega) \quad i = 1, \cdots, m \tag{9}$$

Several existing algorithms for the solution of general convolutive or instantaneous mixture problems have exploited the 2-D structure of expressions equivalent or similar to equation (9), e.g. by time-frequency masking [25] or joint diagonalization [24; 1]. However, the 2-D representation of a 1-D random process is redundant and sometimes may conceal the structure of the underlying data due to interference [10]. Furthermore, the increased amount of data often results in prohibitive computational costs for large-scale problems.

This redundancy can be avoided by projecting the WVS back onto several 1-D random processes. The most simple projections are obtained by computing the first- and zero-order moments of equation (9). These moments can be computed analytically making use of equations (5) and (7), resulting in:

$$E\{|\mathcal{F}x_i|^2\} = \int W_{x_i}(t,\omega)dt = \sum_j^n |\alpha|_{ij}^2 \int W_{s_j}(t - \tau_{ij}, \omega)dt = \sum_j^n |\alpha|_{ij}^2 E\{|\mathcal{F}s_j|^2\} \tag{10}$$

and analogously

$$E\{|\mathcal{F}x_i(\omega)|^2\} \cdot t_{x_i}(\omega) = \sum_j^n \left( |\alpha|_{ij}^2 \cdot E\{|\mathcal{F}s_i|^2\} \cdot \left[ t_{s_j}(\omega) + \tau_{ij} \right] \right). \tag{11}$$

Equation (10) defines an instantaneous mixture problem with non-negativity constrains. Such problems can be treated with standard nonnegative matrix factorization [14] or ICA [11; 12] approaches. After solving equation (10), the remaining unknowns in equation (11) are the delays and the group delay (complex phase). By successive solution of these two equations the expected values of all unknown parameters can be determined. So instead of resolving (9) it is sufficient to solve (10) and (11) by successive iteration. This suggests the following two-step algorithm to estimate the unknowns in the model (1).

## 2.4 Two-step algorithm

The last result implies the following algorithm for the estimation of the sources, delays, and the mixing matrix in (1):

1. Compute $|\mathcal{F}x_i|^2$ and solve

$$|\mathcal{F}x_i|^2(\omega) = \sum_{j}^{n} |\alpha|_{ij}^2 |\mathcal{F}s_j|^2(\omega) \tag{12}$$

   e.g. using non-negative matrix factorization. For our implementation we applied an algorithm for Bayesian non-negative ICA [11].

2. Initialize $\tau_{ij} = 0$ and iterate the following steps:

   (a) Numerically solve :

$$|\mathcal{F}x_i(\omega)|^2 \cdot \frac{\partial}{\partial \omega} \arg\{\mathcal{F}x_i\} = \sum_{j}^{n} |\alpha|_{ij}^2 \cdot |\mathcal{F}s_i|^2 \cdot \left[ \frac{\partial}{\partial \omega} \arg\{\mathcal{F}s_j\} + \tau_{ij} \right] \tag{13}$$

   for the term $\frac{\partial}{\partial \omega} \arg\{\mathcal{F}s_j\}$. Integrate the solution to obtain $\mathcal{F}s_j$.

   (b) Exploiting the knowledge of the sources $s_j$ it is possible to update the mixing and the delay matrix through optimization of the following cost function, which is derived from (1), where $\mathbf{S}(\vec{\tau_j}) = (s_k(t_i - \tau_{jk}))_{i,k}$, $\mathbf{A}_j = (\alpha_{ij})_i$:

$$[\widehat{\vec{\tau_j}}, \widehat{\mathbf{A}_j}] = \operatorname{argmin}_{[\vec{\tau_j}, \mathbf{A}_j]} \|x_j - \mathbf{A}_j \cdot \mathbf{S}(\vec{\tau_j})\|_2 \tag{14}$$

   This minimization is accomplished following [21], assuming uncorrelatedness for the sources and independence of the time delays.

   (c) Update $\tau_{ij}$ and go back to (a), until convergence is achieved.

## 3 Test data sets

For comparing the novel algorithm with other related methods we used two different types of data sets. A first set of data was generated artificially by mixing different sound sources, varying the mixing weights and the delays. This data was non-periodic and enabled us to validate the accuracy of the reconstruction of the source signals. The second data set were human movement trajectories of emotional gaits that were recorded using motion capture. The gait trajectories were periodic and served for testing the suitability of the new method for extracting biologically interpretable movement components.

Our first data set consisted of synthetically generated delayed mixtures generated from segments of speech and sound signals taken from an ICA benchmark data set described in [6]. The data basis contained in total 14 signals, with a length of 8000 time points each. In order to obtain statistically representative results, data sets were recomputed 20 times with random selection of the source signals, and/or of the mixing and delay matrices. Three types of mixtures were generated:

 (I) Mixtures of 2 source segments with random mixing and delay matrices ($2 \times 2$), each resulting in two simulated signals $x_{1,2}$. This data set was used to compare the new method with PCA and (fast) ICA [12]. Data set (I) was included to show that the new algorithm is also able to address the even-determined case ($n = m$).

 (II) Mixtures of 2 randomly selected segments from the speech data basis using the constant mixing matrix $\mathbf{A} = [1, 2; 3, 1; 10, 5; 1, 2; 1, 1]$ and the constant delay matrix $\mathbf{T} = (\tau_{ij})_{ij} = [0, 4000; 2500, 5000; 100, 200; 1, 1; 500, 333]$. This data set was used to compare the new method with PCA and ICA, and the SOBIDS algorithm [1], which requires at least twice as many signals as sources. Data set (II) with fixed mixing and delay matrices was included since completely random generation sometimes produced degenerated anechoic mixtures (instantaneous mixtures or ill-conditioned mixing matrices).

(III) A third data set was generated by mixing two randomly selected source segments with random mixture matrices and random delay matrices.

To compare the performance of the different algorithms we used a performance measure $M$ that was defined by the maximum of the cross-correlations between extracted sources, $s_{\text{extract},j}$, and original sources $s_{\text{orig},j}$ (after appropriate matching of the individual sources, since the recovered sources are not ordered in a specific manner):

$$M = (1/n) \sum_{j=1}^{n} \max_{\tau} |E\{s_{\text{extract},j}(t) \cdot s_{\text{orig},j}(t+\tau)\}|$$

The second test data set consisted of movement trajectories of human actors walking neutrally, or with different emotional styles (happy, angry, sad and fearful). The movements were recorded using a VICON 612 motion capture system with 7 cameras, obtaining the 3D positions of 41 passive markers on the bodies of the actors. We recorded trajectories from 13 lay actors, repeating each walking style three times per actor. A hierarchical kinematic body model (skeleton) with 17 joints was fitted to the marker positions, and joint angles were computed. Rotations between adjacent body segments were described as Euler angles, defining flexion, abduction and rotation about the connecting joints. The data for the unsupervised learning procedure included only the flexion angles of the hip, knee, elbow, shoulder and the clavicle, since the other angles had relatively high noise levels. From each trajectory only one gait cycle was extracted, which was time normalized. This resulted in a data set with 1950 samples with a length of 100 time points each.

## 3.1 Results

**Delayed mixtures of sound sources:** Figure 1 shows the results for the extraction of the sound sources from the data sets (I)-(III). The bar plots show means and standard deviations of the performance measure $M$ over twenty simulations. On all data sets our new method shows an overall performance measure above $80\%$, while PCA and ICA show performances between $50$ and $60\%$. The SOBIDS algorithm reaches a performance level of $72\%$.

For a more accurate statistical comparison of the different methods we used a one-way repeated measure ANOVA for the measure $M$. There was a significant effect for all three data sets. Post-hoc comparison using the Least Significant Difference (LSD) [18] revealed that our new method was significantly better than PCA and ICA for all three data sets. Our method significantly outperformed the SOBIDS algorithm for data set (III). For data set (II) the difference between these two algorithms was not significant, due to the increased overall variability in this data set.

The better performance of our algorithm compared to PCA and ICA results from the appropriate modeling of time delays. The better performance compared to the SOBIDS algorithm might be explained by the fact that this algorithm requires the assumption of small delays.

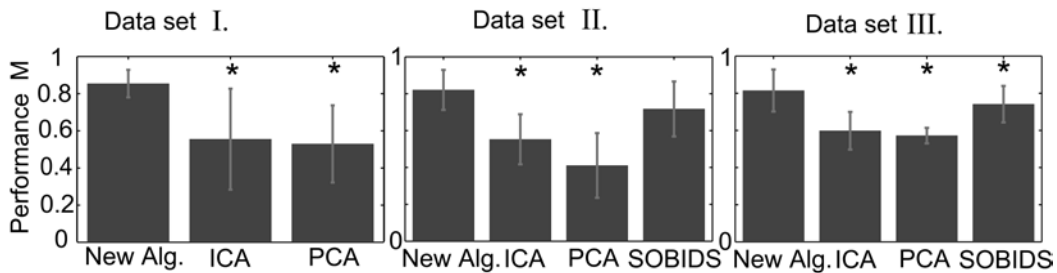

Figure 1: Comparison of different blind source separation algorithms for synthetic mixtures of sound signals with delays (data sets I-III, see text). The stars indicate significant ($p < 0.001$) differences compared to the new algorithm.

**Human gait trajectories:** With the second data set we tested whether the proposed novel algorithm is suitable for the extraction of interpretable source signals from human movement data. By performing normal ICA separately on individual joint trajectories and comparing the extracted sources, we had observed before that such sources are often very similar except for time shifts between different joints. This motivates the hypothesis that (1) might provide an appropriate generative model for such gait trajectories.

This hypothesis is confirmed by the data presented in Figure 2 that shows the approximation quality (explained variance) for different numbers of extracted sources and comparing four different algorithms: PCA, (fast) ICA [12], Bayesian positive ICA [11], SOBIDS, and the new algorithm. The new method outperforms all other methods, and in particular the methods without time delays. Specifically, the new algorithm is capable of approximating 97% of the trajectory data with only 3 sources, while PCA and ICA require more than 6 sources to achieve the same level of accuracy.

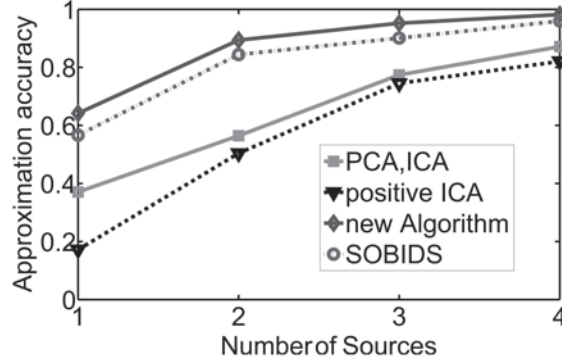

Figure 2: Comparison of different blind source separation algorithms. Explained variance is shown for different numbers of extracted sources.

In order to test whether the novel algorithm results in source signals that provide useful interpretations of biological data we modeled all trajectories in our gait data sets by linear superpositions of the extracted sources and analyzed the resulting mixture matrices $\mathbf{A}$. To extract weight components that are specific for individual emotional styles we modeled the mixture matrices applying *sparse linear regression*. The (vectorized) weights of the individual gait trajectories for emotion $j$, defining the vector $\mathbf{a}_j$, were approximated by the sum of a component $\mathbf{a}_0$ (containing the weights that characterize neutral walking) and an emotion-specific contribution. Formally, this multi-linear regression model can be written as

$$\mathbf{a}_j \approx \mathbf{a}_0 + \mathbf{C} \cdot \mathbf{e}_j \, , \tag{15}$$

where $\mathbf{C}$ is a weight matrix that determines the emotion-specific contributions to the mixing weights. Its columns are given by the differences between the weights for the different emotional styles (happy, sad, fearful and angry) and the weights for neutral walking. In order to obtain easily interpretable results, the matrix $\mathbf{C}$ was sparsified by $L_1$ norm minimization. The solution of the linear regression problem was obtained by minimizing the following cost function (with $\gamma > 0$) using quadratic programming:

$$E(\mathbf{C}) = \sum_j \|\mathbf{a}_j - \mathbf{a}_0 - \mathbf{C} \cdot \mathbf{e}_j\|^2 + \gamma \sum_{i,j} |C_{ij}| \tag{16}$$

This regression basically computes the mean differences between the weights for neutral walking and for emotional walking. The sparsification separates automatically important and less important features. The concentration of the variance into a few important predictors simplifies the interpretation. Figure 3 shows a gray level plot of the matrix $\mathbf{C}$, illustrating the weight differences compared to neutral walking for the four different emotional styles and the different joint angles. Positive elements of the matrix indicate cases where the joint amplitudes for the emotional gait are increased compared to normal walking. Negative elements are indicated by white triangles in the lower left corner of the individual cells of the plot. They correspond to cases where the joint angle amplitudes for the emotional walk are reduced compared to normal walking. The $+$ and $-$ signs in the figure summarize data from psychophysical experiments that have investigated kinematic features that were important for the perception of emotional gaits [17; 23]. Plus signs indicate cases where (perceived) increases of the joint amplitudes compared to normal walking were correlated with the perception of the corresponding emotion, and minus signs to cases where a (perceived) reduction of the joint angle amplitudes was correlated with the perception of the corresponding emotion.

Comparison between these psychophysical results and the elements of the matrix $\mathbf{C}$ (Figure 3a) shows a very close match between the weight changes and the features that are important for the perception of emotions from gaits. This implies that the novel algorithm extracts features that can

be meaningfully interpreted in a biological sense. Figure 3b shows the results of the same analysis for sources that had been extracted with PCA, matching the numbers of non-zero elements of the estimated matrix **C**. For gait trajectories the source signals by PCA and ICA are virtually identical. Therefore, results in Figure 3 would be unchanged for ICA. In either case, the match is significantly worse than for the sources extracted with the novel algorithm (panel a). In addition, the signs of the matrix elements often do not match the signs of the amplitude changes in the psychophysical experiments. This implies that the new algorithm extracts spatio-temporal components from human gait data that are more easily interpretable than components extracted with PCA.

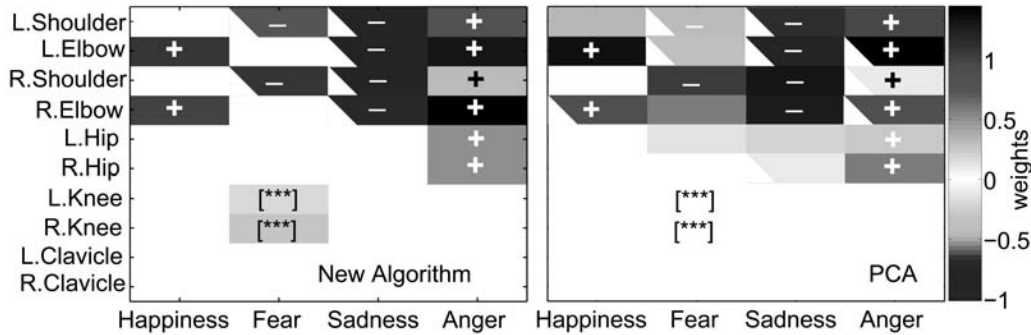

Figure 3: Elements of the weight matrix **C**, encoding emotion-specific deviations from neutral walking, for different degrees of freedom. Negative elements are indicated by white triangles in the lower left corners of the cells. Kinematic features that have been shown to be important for the perception of emotions from gait in psychophysical experiments are indicated by the plus and minus signs. (Details see text.)

## 3.2   Conclusion

We present a new algorithm for the solution of over-determined blind source separation problems for mixtures of sources with delays. The proposed method has been derived by application of a time-frequency transformation to the mixture model, resulting in a two-step algorithm that combines positive ICA with another iterative optimization step. We demonstrate that the developed algorithm outperforms other source separation algorithms with and without time delays on synthetic data sets defined by delayed mixtures of speech signals, and also on real data sets obtained by motion capture of human full-body movements. For human movements we also demonstrate that, at least for the case of human gait, the new algorithm provides a more compact and interpretable representations than the alternative methods we tested. To our knowledge the proposed algorithm is the first one that solves over-determined delayed mixing problems without specific additional assumptions about the structure of the delay matrix, e.g. limited sizes of the delays. In contrast to nonnegative matrix factorization with delays [2], the proposed method is applicable to non-positive signals and sources. Future work will focus on testing the algorithm with a broader range of data sets, also including particularly non-periodic human movements. In addition, it seems possible to extend the proposed method for multi-dimensional translation vectors (delays), making it applicable for the learning of translation-invariant features in two-dimensional images.

### Acknowledgments

This work was supported by HFSP, DFG, the Volkswagenstiftung and the EU FP6 Project 'COBOL'. We thank C.L. Roether for help with the trajectory acquisition and the psychological interpretation of the data, and W. Ilg for support with the motion capturing.

## References

[1] J. Ashtar, et al (2004) A novel approach to blind separation of delayed sources in linear mixtures. 7th Semester Signal Processing, Aalborg University.

[2] A. d'Avella, E. Bizzi (2005) Shared and specific muscle synergies in natural motor behaviors. Proc Natl Acad Sci U S A **102(8)** 3076-3081.

[3] A.J. Bell, T.J. Sejnowski (1995) An information-maximization approach to blind separation and blind deconvolution. Neural Computation **7** 1129-1159.

[4] P. Bofill (2003) Underdetermined blind separation of delayed sound sources in the frequency domain. Neurocomputing **Vol. 55** 627-641.

[5] A. Celik, et al (2005) Gradient Flow Independent Component Analysis in Micropower VLSI. Advances in Neural Information Processing Systems **18**, 187-194.

[6] A. Cichocki, S. Amari, (2002) *Adaptive Blind Signal and Image Processing.* John Wiley, Chichester (2002.)

[7] L. Cohen (1995) *Time-Frequency Analysis.* Englewood Cliffs, NJ. PrenticeHall.

[8] B. Emile, P. Comon (1998) Estimation of time delays between unknown colored signals. Signal Processing **69** 93–100.

[9] P.D. O'Grady, B.A. Pearlmutter, S.T. Rickard (1982) Survey of sparse and non-sparse methods in source separation. International Journal of Imaging Systems and Technology (IJIST), special issue on blind source separation and deconvolution in imaging and image processing (**15**).

[10] F. Hlawatsch, P. Flandrin (1997) The Interference Structure of the Wigner Distribution and Related Time-Frequency Signal Representations. *The Wigner Distribution -Theory and Applications in Signal Processing*. Amsterdam: Elsevier, 59-133.

[11] P. Hojen-Sorensen, O. Winther, L. Hansen (2002) Mean field approaches to independent component analysis. Neural Computation **14** 889-918.

[12] A. Hyvärinen, E.O., (1997) A fast fixed-point algorithm for independent component analysis. Neural Computation **9** 1483-1492.

[13] Y. Ivanenko, R. Poppele, F. Lacquaniti (2004) Five basic muscle activation patterns account for muscle activity during human locomotion. J Physiol. **556(Pt1)** 267-282.

[14] D.D. Lee, H.S. Seung (1999) Learning the parts of objects by Non-Negative Matrix Factorization. Nature **401**.

[15] W. Martin (1982) Time-frequency analysis of random signals. Proc. IEEE Int. Conf. on Acoust., Speech and Signal Processing. 1325-1328.

[16] G. Matz, F. Hlawatsch (2003) Wigner distributions (nearly) everywhere: Time-frequency analysis of signals, systems, random processes, signal spaces, and frames. Signal Processing, special section "From Signal Processing Theory to Implementation" on the occasion of the 65th birthday of W. Mecklenbräuker **83** 1355-1378.

[17] M. de Meijer (1989) The contribution of general features of body movement to the attribution of emotions. Journal of nonverbal behaviour **13** 247-268

[18] R. G. Miller Jr. (1981) *Simultaneous Statistical Inference*. Springer, New York, NY, 2nd edition.

[19] R. Vigário, V. Jousmäki, M. Hämäläinen, R. Hari, E.Oja (1998) Independent component analysis for identification of artifacts in magnetoencephalographic recordings. Advances in Neural Information Processing Systems **10** 229-235.

[20] R. Roy, T. Kailath (1989) ESPRIT—Estimation of signal parameters via rotational invariance techniques. IEEE Trans. Acoust., Speech, Sig. Proc. **37** 984-995.

[21] A. Swindelhurst (1998) Time delay and spatial signature estimation using known asynchronous signals. IEEE Trans. on Sig. Proc. **ASSP-33,no. 6** 1461-1470.

[22] K. Torkkola (1996) Blind separation of delayed sources based on information maximization. ICASSP'96 3509-3512.

[23] H. G. Wallbott (1998) Bodily expression of emotion. European Journal of Social Psychology **28** 879-896.

[24] A. Yeredor (2003) Time-delay estimation in mixtures. Acoustics, Speech, and Signal Processing **5** 237-240.

[25] Ö. Yýlmaz, S.Rickard (2004) Blind Separation of Speech Mixtures via Time-Frequency Masking. IEEE Transactions On Signal Processing **52** 1830-1847.
